# Spikernels:
# Embedding Spiking Neurons
# in Inner-Product Spaces

**Lavi Shpigelman**[†§] **Yoram Singer**[†] **Rony Paz**[§♭] **Eilon Vaadia**[§♭]
[†]School of computer Science and Engineering
[§]Interdisciplinary Center for Neural Computation
[♭]Dept. of Physiology, Hadassah Medical School
The Hebrew University Jerusalem, 91904, Israel
{shpigi,singer}@cs.huji.ac.il
{ronyp,eilon}@hbf.huji.ac.il

## Abstract

Inner-product operators, often referred to as *kernels* in statistical learning, define a mapping from some input space into a feature space. The focus of this paper is the construction of biologically-motivated kernels for cortical activities. The kernels we derive, termed Spikernels, map spike count sequences into an abstract vector space in which we can perform various prediction tasks. We discuss in detail the derivation of Spikernels and describe an efficient algorithm for computing their value on any two sequences of neural population spike counts. We demonstrate the merits of our modeling approach using the Spikernel and various standard kernels for the task of predicting hand movement velocities from cortical recordings. In all of our experiments all the kernels we tested outperform the standard scalar product used in regression with the Spikernel consistently achieving the best performance.

## 1 Introduction

Neuronal activity in primary motor cortex (MI) during multi-joint arm reaching movements in 2-D and 3-D [1, 2] and drawing movements [3] has been used extensively as a test bed for gaining understanding of neural computations in the brain. Most approaches assume that information is coded by firing rates, measured on various time scales. The tuning curve approach models the average firing rate of a cortical unit as a function of some external variable, like the frequency of an auditory stimulus or the direction of a planned movement. Many studies of motor cortical areas [4, 2, 5, 3, 6] showed that while single units are broadly tuned to movement direction, a relatively small population of cells (tens to hundreds) carries enough information to allow for accurate prediction. Such broad tuning can be found in many parts of the nervous system, suggesting that computation by distributed populations of cells is a general cortical feature. The population-vector method [4, 2] describes each cell's firing rate as the dot product between that cell's preferred direction and the direction of hand movement. The vector sum of preferred directions, weighted by the measured firing rates is used both as a way of understanding what the cortical units encode and as a means for estimating the velocity vector.

Several recent studies [7, 8, 9] propose that neurons can represent or process multiple parameters simultaneously, suggesting that it is the dynamic organization of the activity in neuronal populations that may represent temporal properties of behavior such as the computation of transformation from 'desired action' in external coordinates to muscle activation patterns. Some studies

[10, 11, 12] support the notion that neurons can associate and dissociate rapidly to functional groups in the process of performing a computational task. The concepts of simultaneous encoding of multiple parameters and dynamic representation in neuronal populations, could together explain some of the conundrums in motor system physiology. These concepts also invite usage of increasingly complex models for relating neural activity to behavior. Advances in computing power and recent developments of physiological recording methods allow recording of ever growing numbers of cortical units that can be used for real-time analysis and modeling. These developments and new understandings have recently been used to reconstruct movements on the basis of neuronal activity in real-time in an effort to facilitate the development of hybrid brain-machine interfaces that allow interaction between living brain tissue and artificial electronic or mechanical devices to produce brain controlled movements [13, 6, 14, 15, 11, 16, 17]. Current attempts at predicting movement from cortical activity rely on modeling techniques such as cosine-tuning estimation (pop. vector) [18], linear regression [15, 19] and artificial neural nets [15] (though this study reports getting better results by linear regression). A major deficiency of standard approaches is poor ability to extract the relevant information from monitored brain activity in an efficient manner that will allow reducing the number of recorded channels and recording time.

The paper is organized as follows. In Sec. 2 we describe the problem setting that this paper is concerned with. In Sec. 3 we introduce and explain the main mathematical tool that we use, namely, the kernel operator. In Sec. 4 we discuss the design and implementation of a biologically-motivated kernel for neural activities. We report experimental results in Sec. 5 and give conclusions in Sec. 6.

## 2   Problem setting

Consider the case where we monitor instantaneous spike rates from $d$ cortical units during physical motor behavior of a subject. Our goal is to learn a predictive model of some behavior parameter with the cortical activity as the input. Formally speaking, let $\mathbf{s} \in \mathbb{R}^{q \times T}$ be a sequence of instantaneous firing rates from $q$ cortical units consisting of $T$ samples altogether. We use $\mathbf{s}, \mathbf{t}$ to denote sequences of firing rates and denote by $len(\mathbf{s})$ the length of a sequence $\mathbf{s}$. Let $\mathbf{s}_i$ be the $i$th sample (i.e. instantaneous firing rates) of a sequence $\mathbf{s}$. We also use $\mathbf{s}x$ to denote the concatenation of $\mathbf{s}$ with one more sample $x$. We refer to the instantaneous firing rate of a unit $k$ by $x_k$. We also need to employ a notation for sub-sequences. The $t$-long prefix $\mathbf{s}$ is denoted $\mathbf{s}_{1:t}$. Finally, throughout the work we need to examine a substrings of sequences. We denote by $\mathbf{i}$ a vector of indices into the sequence $\mathbf{s}$ where $\mathbf{i} = (i_1, i_2, \ldots, i_n)$ and $1 \le i_1 \le i_2 \le \ldots \le i_n \le len(\mathbf{s})$.

We also need to introduce some notation for target variables we would like to predict. Let $y \in \mathbb{R}^T$ denote some parameter of the movement that we would like to predict (e.g. the movement velocity in the $x$ direction, $v_x$ ). Our goal is to learn an approximation $\widehat{y}_t$ of the form $f : \mathbb{R}^{q \times T} \to \mathbb{R}^T$ from neural firing rates to movement parameter. In general, information about movement can be found in neural activity both before and after the time of movement itself. Our plan, though, is to design a model that can be used for controlling a neural prosthesis. We will therefore confine ourselves to causal predictors that use $\mathbf{s}_{1:t}$ to predict $y_t$. We therefore would like to make $\widehat{y}_t = f(\mathbf{s}_{1:t})$ as close as possible (in a sense that is explained in the sequel) to $y_t$.

## 3   Kernel methods for regression

A major mathematical notion employed in this paper is kernel operators. Kernel operators allow algorithms whose interface to the data is limited to scalar products to employ complicated premappings of the data into feature spaces by use of kernels. Formally, a kernel is an inner-product operator $K : X \times X \to \mathbb{R}$ where $X$ is some arbitrary vector space. An explicit way to describe $K$ is via a mapping $\phi : X \to \mathcal{H}$ from $X$ to an inner-products space $\mathcal{H}$ such that $K(x, x') = \phi(x) \cdot \phi(x')$. Given a kernel operator we can use it to perform various statistical learning tasks. One such task is support vector regression (SVR) [20] which attempts to find a regression function for target values that is linear if observed in the (typically very large) feature space mapped by the kernel. We give here a brief description of SVR for the the sake of clarity.

Support Vector Regression minimizes Vapnik's [21] $\varepsilon$-insensitive loss function $|y - f(\mathbf{x})|_\varepsilon = \max\{0, |y - f(\mathbf{x})| - \varepsilon\}$ which defines a hyperplane with width $\varepsilon$ around the estimate. Examples that fall within it's boundaries are considered well estimated and do not contribute to the error. Examples outside the tube contribute linearly to the loss. Say $\phi(\mathbf{x})$ is the feature vector implemented by kernel $K(\cdot, \mathbf{x})$. To estimate a linear (linear in feature space) regression $f(\mathbf{x}) = (\mathbf{w} \cdot \phi(\mathbf{x})) + \mathbf{b}$ with precision $\varepsilon$, one minimizes

$$\frac{1}{2} \|\mathbf{w}\|^2 + C \sum_{i=1}^{m} |y_i - f(\phi(\mathbf{x}_i))|_\varepsilon$$

This can be written as a constrained minimization problem

$$\text{minimize} \quad \tau(\mathbf{w}, \xi, \xi^*) = \frac{1}{2} \|\mathbf{w}\|^2 + C \sum_{i=1}^{m} (\xi_i + \xi_i^*)$$

$$\text{subject to} \quad (\mathbf{w} \cdot \phi(\mathbf{x}_i) + b) - y_i \leq \varepsilon + \xi_i$$
$$y_i - (\mathbf{w} \cdot \phi(\mathbf{x}_i) + b) \leq \varepsilon + \xi_i^*$$
$$\xi_i, \xi_i^* \geq 0$$

By switching to the dual problem of this optimization problem, it is possible to incorporate the kernel function, achieving a mapping that may not be feasible by calculating (possibly infinite) feature vectors $\phi(\mathbf{s})$. For $C > 0$, $\varepsilon \geq 0$ chosen a-priori, the dual problem is

$$\text{maximize} \quad W(\alpha, \alpha^*) = \quad -\varepsilon \sum_{i=1}^{m}(\alpha_i^* + \alpha_i) + \sum_{i=1}^{m}(\alpha_i^* - \alpha_i)y_i$$

$$-\frac{1}{2} \sum_{i,j=1}^{m} (\alpha_i^* - \alpha_i)(\alpha_j^* - \alpha_j) k(\mathbf{x}_i, \mathbf{x}_j)$$

$$\text{subject to} \quad \forall i \in \{1, \ldots, m\}: \quad \alpha_i, \alpha_i^* \in [0, C] \text{ and } \sum_{i=1}^{m}(\alpha_i - \alpha_i^*) = 0$$

The solution of the regression estimate takes the form

$$f(\mathbf{x}) = \sum_{i=1}^{m}(\alpha_i^* - \alpha_i)k(\mathbf{x}_i, \mathbf{x}) + b$$

In summary, SVM regression solves a quadratic optimization problem to find a hyperplane in the kernel induced feature space that best estimates the data for an $\varepsilon$-insensitive linear loss function.

## 4  Spikernels

The quality of SVM learning is highly dependent on how the data is embedded in the feature space via the kernel operator. For this reason, several studies have been devoted lately to developing new kernels [22, 23, 24]. In fact, for classification problems, a good kernel would render the work of the classification algorithm trivial. With this in mind, we develop a kernel for neural spiking activity.

### 4.1  Motivation

Our goal in developing a kernel for spike trains is to map similar patterns to nearby areas of the feature space. Current methods for predicting response variables from neural activities use standard linear regression techniques (see for instance [15]) or or even replace the time pattern with mean firing rates. A notable example is the population vector method [18]. Other approaches use off-the-shelf learning algorithms, intended for general purpose. In the description of our kernel we attempt to capture some well accepted notions on similarities between spike trains. We make the following assumptions regarding similarities between spike patterns:

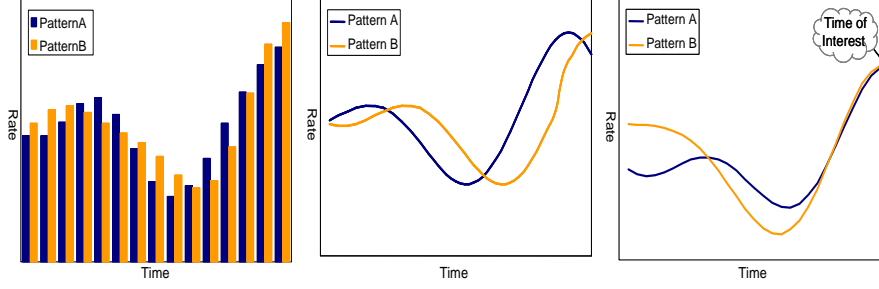

Figure 1: Illustrative examples of pattern similarities. Left: bin-by-bin comparison yields small differences. Middle: patterns with large bin-by-bin differences that can be eliminated with some time warping. Right: patterns whose suffix (time of interest) is similar and prefix is different.

• The most commonly made assumption is that similar firing patterns may have small differences in a bin-by-bin comparison. This type of variation is due to inherent noise of any physical system but also responses to external factors that were not recorded and are not directly related the to the task performed. On the left-hand side of Fig. 1 we show an example of two patterns that are bin-wise similar though clearly not identical.

• A cortical population may display highly specific patterns to represent specific information. It is conceivable that some features of external stimuli are represented by population dynamics that would be best described as 'temporal' coding.

• Two patterns may be quite different in a simple bin-wise comparison but if they are aligned by some non-linear time distortion or shifting, the similarity becomes apparent. An illustration of such patterns is given in the middle plots of Fig. 1. In comparing patterns we would like to induce a higher score when the time-shifts are small.

• Patterns that are associated with identical values of an external stimulus at time $t$ may be similar at that time but very different at $t \pm \Delta$ when values of the external stimulus for these patterns are no longer similar (as illustrated on the right-hand-side of Fig. 1).

### 4.2 Kernel definition

We describe the kernel by specifying the features that make up the feature space. Our construction of the feature space builds on the work of Lodhi et al. [24]. First, we need to introduce a few more notations. Let $\mathbf{s}$ be a sequence of length $l = len(\mathbf{s})$. The set of all possible $n$-long index vectors defining a sub-sequence of $\mathbf{s}$ is $\mathbf{I_{n}}_{,l} = \{\mathbf{i} : \mathbf{i} \in \mathbb{Z}^n \, 1 \leq \mathbf{i}_1 < \ldots < \mathbf{i}_n \leq len(\mathbf{s})\}$. Also, let $d(\alpha, \beta)$ denote a bin-wise distance over a pair of samples (firing rates). We also overload notation and denote by $d(\mathbf{s_i}, \mathbf{u}) = \sum_{k=1}^{n} d(\mathbf{s}_{i_k}, \mathbf{u}_k)$ a distance between sequences. The sequence distance is the sum over the samples constituting the two sequences. Let $\mu, \lambda \in (0, 1)$. The $\mathbf{u}$ component of our (infinite) feature vector $\phi(\mathbf{s})$ is defined as,

$$\phi_{\mathbf{u}}(\mathbf{s}) = C^{\frac{n}{2}} \sum_{\mathbf{i} \in \mathbf{I}_{n,len(\mathbf{s})}} \mu^{d(\mathbf{s_i}, \mathbf{u})} \lambda^{len(\mathbf{s}) - \mathbf{i}_1} \quad , \tag{1}$$

where and $C$ is a normalization constant that simplifies the calculation and and $\mathbf{i}_1$ is the first index of $\mathbf{i}$. In words, $\phi_{\mathbf{u}}(\mathbf{s})$ is a sum over all n-long sub-sequences of $\mathbf{s}$. Each sub-sequence is compared to $\mathbf{u}$ (the feature coordinate) and is weighted up according to its similarity to $\mathbf{u}$. In particular, part of the weight of each sub-sequence of $\mathbf{s}$ reflects how concentrated the sub-sequence is toward the end of $\mathbf{s}$. Put another way, the entry indexed by $\mathbf{u}$ measures how close $\mathbf{u}$ is to the time series $\mathbf{s}$ near its end.

This definition seems to fit our assumptions on neural coding for the following reasons:

• It allows for complex patterns: small values of $\lambda$ and $\mu$ (or concentrated $d$ measures) mean that each coordinate $\mathbf{u}$ tends toward being either 1 or 0 depending whether $\mathbf{u}$ is almost identical to a suffix of $\mathbf{s}$ or not.

- Patterns that are piece-wise similar to $\mathbf{u}$ contribute to the $\mathbf{u}$ feature coordinate with a weight that decays as the sample-by-sample comparison between the sequences grows large.

- We allow gaps in the indexes defining sub-sequences, thus, allowing for time warping.

- Patterns that begin further from the required prediction time are penalized by an exponentially decaying weight.

### 4.3 Efficient kernel calculation

he definition of $\phi$ given by Eq. (1) requires the manipulation of an infinite feature space. Straight-forward calculation of the feature values and performing the induced inner-product is clearly impossible. Based on ideas from [24] we developed an indirect method for evaluating the kernel through a recursion which can be performed efficiently using dynamic programing. We now describe the recursion.

Denote by $x \in \mathbb{R}^q$ the last entry in the sequence $\mathbf{s}x \in \mathbb{R}^{d \times len(\mathbf{s}x)}$. We now describe two recursive equations for $\phi$ with respect to the length of the time series and the sub-sequence length. Due to the lack of space we skip some of the algebraic manipulations that are needed to derive the recursions. The first equation is

$$\phi_{\mathbf{u}}(\mathbf{s}x) \;=\; \lambda \phi_{\mathbf{u}}(\mathbf{s}) + \lambda \mu^{d(\mathbf{u}_n, x)} C^{\frac{1}{2}} \phi_{\mathbf{u}_{1:n-1}}(\mathbf{s}) \tag{2}$$

Eq. (2) simply separates the sum over sub-sequences of $\mathbf{s}$ into two subsets: one where $x$ is not specified by the index vectors and the latter where $\mathbf{i}_n$ specifies $x$. The second recursive equation for $\phi$ is, again, with respect to both the length of the sub-sequence ($\mathbf{u}$) and the length of the sequence $\mathbf{t}$,

$$\phi_{\mathbf{u}}(\mathbf{t}) \;=\; C^{\frac{1}{2}} \sum_{j=1}^{len(\mathbf{t})} \mu^{d(\mathbf{u}_n, \mathbf{t}_j)} \lambda^{len(\mathbf{t}) - j + 1} \phi_{\mathbf{u}_{1:n-1}}(\mathbf{t}_{1:j-1}) \tag{3}$$

The last equation simply states that the feature is a sum over all possible values of $\mathbf{i}_n$. Note that for $j < n$, $\mathbf{I}_{n,j}$ is empty. Eqs. (2) and (3) are now used for computing the recursion equation for $K$:

$$K_n(\mathbf{s}x, \mathbf{t}) = \int_{\mathbb{R}^{q \times n}} \phi_{\mathbf{u}}(\mathbf{s}x) \phi_{\mathbf{u}}(\mathbf{t}) \, d\mathbf{u}$$

We plug Eq. (2) into $\phi_{\mathbf{u}}(\mathbf{s}x)$ and plug Eq. (3) into $\phi_{\mathbf{u}}(\mathbf{t})$. Using algebraic manipulations we replace integrals over scalar products of $\phi$ by the proper kernels and get the following recursive function,

$$K_n(\mathbf{s}x, \mathbf{t}) \;=\; \lambda K_n(\mathbf{s}, \mathbf{t}) + C \sum_{j=1}^{len(\mathbf{t})} \lambda^{len(\mathbf{t}) - j + 2} K_{n-1}(\mathbf{s}, \mathbf{t}_{1:j-1}) \int_{\mathbb{R}^d} \mu^{d(\mathbf{u}_n, x)} \mu^{d(\mathbf{u}_n, \mathbf{t}_j)} \, d\mathbf{u}_n \tag{4}$$

The initial conditions are:

$$\forall \mathbf{s} \in \mathbb{R}^{d \times len(\mathbf{s})}, \mathbf{t} \in \mathbb{R}^{d \times len(\mathbf{t})} \quad K_0(\mathbf{s}, \mathbf{t}) \;=\; 1$$
$$\text{if } \min\{len(\mathbf{s}), len(\mathbf{t})\} < i \quad K_i(\mathbf{s}, \mathbf{t}) \;=\; 0$$

Assuming that the computation time of the integral in Eq. (4) is a constant, computing the entire recursion requires $\mathcal{O}\left(len(\mathbf{s}) \, len^2(\mathbf{t}) n\right)$ time. We can achieve a speed up by a factor of $len(\mathbf{t})$ if we cache the term on the right hand side of Eq.(4) as follows. Define,

$$K'_n(\mathbf{s}x, \mathbf{t}y) \;=\; C \sum_{j=1}^{len(\mathbf{t}y)} \lambda^{len(\mathbf{t}y) - j + 2} K_{n-1}(\mathbf{s}, (\mathbf{t}y)_{1:j-1}) \int_{\mathbb{R}^d} \mu^{d(\mathbf{u}_n, x) + d(\mathbf{u}_n, (\mathbf{t}y)_j)} \, d\mathbf{u}_n \tag{5}$$

Separating the above sum into its two parts (one for $j = len(\mathbf{t}y)$ and one for the rest), and using the definition of $K'$ from Eq.(5) we get the following recursive equation for $K'$,

$$K'_n(\mathbf{s}x, \mathbf{t}y) \;=\; \lambda^2 C K_{n-1}(\mathbf{s}, \mathbf{t}) \int_{\mathbb{R}^d} \mu^{d(\mathbf{u}_n, x) + d(\mathbf{u}_n, y)} \, d\mathbf{u}_n + \lambda K'_n(\mathbf{s}x, \mathbf{t}) \tag{6}$$

$$\forall \mathbf{s} \in \mathbb{R}^{d \times len(\mathbf{s})}, \mathbf{t} \in \mathbb{R}^{d \times len(\mathbf{t})} \quad K'_0(\mathbf{s}, \mathbf{t}) = 1$$
$$\text{if } \min\{len(\mathbf{s}), len(\mathbf{t})\} < i \quad K'_i(\mathbf{s}, \mathbf{t}) = 0$$

Finally, the recursive equation for $K$ is,

$$K_n(\mathbf{s}x, \mathbf{t}) \quad = \quad \lambda K_n(\mathbf{s}, \mathbf{t}) + K'_n(\mathbf{s}x, \mathbf{t}) \ ,$$

yielding an $\mathcal{O}\left(len(\mathbf{s})\, len(\mathbf{t})\, n\right)$ dynamic programing solution for $K_n(\mathbf{s}, \mathbf{t})$.

## 4.4 Spikernel variants.

The kernels defined by Eq.(1) consider only patterns of fixed length ($n$). It makes sense to look at sub-sequences of various lengths. Since a linear combination of kernels is also a kernel, we can define our kernel to be

$$K(s, t) = \sum_{i=1}^{n} q^i K_i(s, t) \ ; \ q > 0 \ .$$

The kernel summation can be interpreted as a concatenation of the feature vectors that these kernels represent. Weighted summation is concatenation of the feature vectors after first multiplying them by the square root of the weights.

Different choices of $d(\alpha, \beta)$ result in kernels that differ in the way two rate values are compared. Say we assign $d(\alpha, \beta)$ to be the squared $\ell_2$ norm: $\|\alpha - \beta\|_2^2 \equiv \sum_{k=1}^{d}\left(\alpha_k - \beta_k\right)^2$, the integral in the kernel recursion Eq.(6) becomes:

$$\int_{\mathbb{R}^d} \mu^{d(u,\alpha)+d(u,\beta)}\, du \quad = \quad \left(\sqrt{\frac{\pi}{-2\ln\mu}}\right)^d \mu^{\frac{1}{2}\|\alpha-\beta\|_2^2}$$

Note that the constant $(\frac{\pi}{-2\ln\mu})^{d/2}$, which has an $n$ fold gain affect on $K$ goes to infinity as $\mu$ goes to 1. This gain results in a kernel whose computation is numerically unstable. However, we can easily cancel it with the constant $C$. Substituting this result back into Eq.(4) we get

$$K'_n(\mathbf{s}x, \mathbf{t}y) \quad = \quad \lambda^2 K_{n-1}(\mathbf{s}, \mathbf{t})\mu^{\frac{1}{2}\|x-y\|_2^2} + \lambda K'_n(\mathbf{s}x, \mathbf{t})$$

We show results for the $\ell_2$ norm.

# 5 Experimental results

**Data collection:** The data used in this work was recorded from the primary motor cortex of a rhesus (Macaca mulatta) monkey (~4.5 kg). The animal's care and surgical procedures accorded with The NIH Guide for the Care and Use of Laboratory Animals (rev. 1996) and with the Hebrew University guidelines supervised by the institutional committee for animal care and use. The monkey sat in a dark chamber, and 8 electrodes were introduced into each hemisphere. The electrode signals were amplified, filtered and sorted (MCP-PLUS, MSD, Alpha-Omega, Nazareth, Israel). The data used in this report includes 31 single units and 16 multi-unit channels (MUA) that were recorded in one session by 16 microelectrodes. The monkey used two planar-movement manipulanda to control 2 cursors (X and + shapes) on the screen to perform center-out task. Each trial begun when the monkey centered both cursors on a central circle for 1.0-1.5s. Either cursor could turn green, indicating the hand to be used in the trial (X for right arm and + for the left). Then, (after an additional hold period of 1.0-1.5s) one of eight targets appeared at a distance of 4 cm from the origin and monkey had to move and reach the target in less than 2s to receive liquid reward. At the end of each session, we examined the activity of neurons evoked by passive manipulation of the limbs and applied intracortical microstimulation (ICMS) to evoke movements. The data presented here was recorded in penetration sites where ICMS evoked shoulder and elbow movements. Penetration locations were verified by MRI (Biospec Bruker 4.7 Tesla).

**Data preprocessing and modeling:** The movements and spike data were preprocessed to create a labeled corpus. We used only the data from trials on which the monkey succeeded in the movement task and examined only the right hand movements. We partitioned the movement and spike trains into $100ms$-long bins to get the spike counts and average hand movement velocities

in each segment. We then normalized the spike counts to achieve a zero mean and a unit variance for each cortical unit. A labeled example $(\mathbf{s}_t, v_t)$ for time $t$ consisted of the $X$ or $Y$ velocity as the target label and the preceding 1 second (i.e. 10 segments) of spike counts from all ($q$) cortical units as the input sequence $\mathbf{s}_t$. In our experiments the number of cortical units $q$ was 47 hence the matrix of spike counts is of size $47 \times 10$.

Each kernel employs a few parameters ($\lambda, \mu, \ldots$) and the SVM regression setup requires setting of two more parameters, ($\varepsilon$ and $C$). Therefore, the learning task is performed in two stages. First, we used cross-validation to choose the best parameters using a validation set. Then, we learned to predict the response variable using SVR. Overall we had 20 minutes of clean cortical recordings of which we used the first 10 minutes as our validation set for tuning the parameters. The second half was used for training and testing. The kernels that we tested are the exponential kernel ($K(\mathbf{s}, \mathbf{t}) = e^{-\gamma(\mathbf{s}-\mathbf{t})^2}$), the homogeneous polynomial kernel ($K(\mathbf{s}, \mathbf{t}) = (\mathbf{s} \cdot \mathbf{t})^d$, $d = 2, 3$), the standard scalar product kernel ($K(\mathbf{s}, \mathbf{t}) = \mathbf{s} \cdot \mathbf{t}$) which boils down to a linear regression, and the Spikernel.

Accuracy results were obtained by performing 5-fold cross-validation for each kernel. The 5 folds were produced by randomly splitting the data into 5 groups: 4 out of the 5 groups were used for training and the rest of the data was used for evaluation. This was process was repeated 5 times by using once each fifth of the data as a test set. We computed the correlation coefficient per fold for each kernel. The per-fold results are shown in Fig. 2A as a scatter plot. Each point compares the Spikernel score versus one of the adversaries. The Spikernel out-performed the rest in every single test set. We found out that predicting the $v_y$ signal was more difficult than predicting the $v_x$ signal. This may be the result of sampling a population of cortical units that are tuned more to the left-right directions. The mean results are summarized in Fig. 2B. The linear regression method (scalar-product kernel) came in last. It seems that both re-mapping the data by standard kernels and by the Spikernel allow for better prediction models. The ordering of the kernels by their mean score is consistent when looking at per-test results, except for the exponential kernel which is out-performed by linear regression in some of the tests.

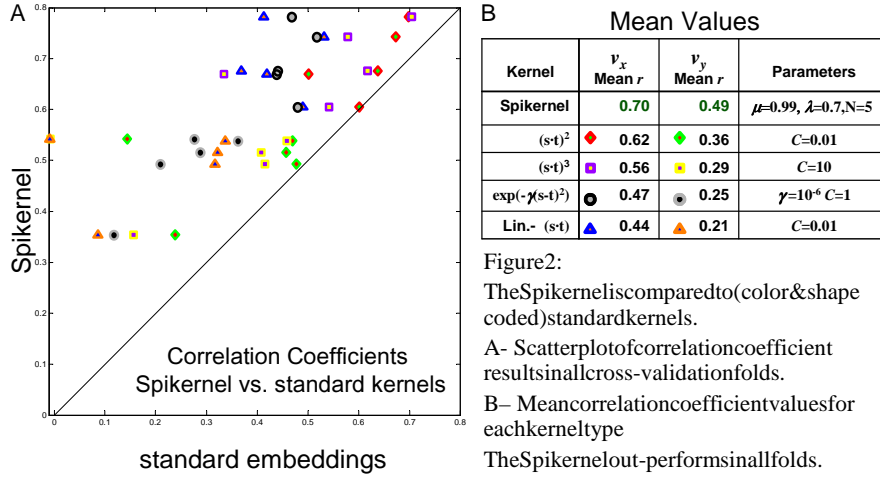

A- Scatter plot of correlation coefficient results in all cross-validation folds. Spikernel vs. standard kernels. X-axis: standard embeddings. Y-axis: Spikernel. Correlation Coefficients.

B — Mean Values

| Kernel | $v_x$ Mean $r$ | | $v_y$ Mean $r$ | | Parameters |
|---|---|---|---|---|---|
| Spikernel | 0.70 | | 0.49 | | $\mu$=0.99, $\lambda$=0.7, N=5 |
| $(\mathbf{s}\cdot\mathbf{t})^2$ | 0.62 | ◆ | 0.36 | ◆ | $C$=0.01 |
| $(\mathbf{s}\cdot\mathbf{t})^3$ | 0.56 | ▣ | 0.29 | ▪ | $C$=10 |
| $\exp(-\gamma(\mathbf{s}\cdot\mathbf{t})^2)$ | 0.47 | ◉ | 0.25 | ◉ | $\gamma$=10$^{-6}$ $C$=1 |
| Lin.- $(\mathbf{s}\cdot\mathbf{t})$ | 0.44 | ▲ | 0.21 | ▲ | $C$=0.01 |

Figure 2:

The Spikernel is compared to (color & shape coded) standard kernels.

A- Scatter plot of correlation coefficient results in all cross-validation folds.

B– Mean correlation coefficient values for each kernel type

The Spikernel out-performs in all folds.

## 6   Summary

In this paper we described an approach based on recent advances in kernel-based learning for predicting response variables from neural activities. On the data we collected, all the kernels we devised outperform the standard scalar product that is used in linear regression. Furthermore, the Spikernel, a biologically motivated kernel operator for spike counts outperforms all the other kernels. Our current research is focused in two directions. First, we are investigating the adaptations of the Spikernel to other neural activities such as Local Field Potentials (LFP). Our second and more challenging goal is to devise statistical learning algorithms that use the Spikernel as

part of a dynamical system that may incorporate bio-feedback. We believe that such extensions are an important and necessary steps toward operational neural prostheses.

**Acknowledgments:** Supported in part by the German-Israeli-Foundation for Scientific Research and Development (GIF) and by the German-Israeli Project Cooperation (DIP) established by BMBF.

# References

[1] Georgopoulos AP, Schwartz AB, and Kettner RE. Neuronal population coding of movement direction. *Science*, 233:1416–1419, 1986.

[2] Apostolos P. Georgopoulus, Ronald E. Kettner, and Andrew B. Wchwartz. Primate motor cortex and free arm movements to visual targets in three-dimensional space. *The Journal of NeuroScience*, 8, August 1988.

[3] Schwartz AB. Direct cortical representation of drawing. *Science*, 265:540–542, 1994.

[4] A. P. Georgopoulus, J.F. Kalaska, and J.T. Massey. Spatial coding of movements: A hypothesis concerning the coding of movement of movement direction by motor cortical populations. *Experimental Brain Research (Supp)*, 7:327–336, 1983.

[5] Daniel W. Moran and Andrew B. Schwartz. Motor cortical representation of speed and direction during reaching. *Journal of Neurophysiology*, 82:2676–2692, 1999.

[6] Mark Laubach, Johan Wessberh, and Miguel A. L. Nicolelis. Cortical ensemble activity increasingly predicts behavior outcomes during learning of a motor task. *Nature*, 405(1), June 2000.

[7] Fu QG, Flament D, Coltz JD, and Ebner TJ. Relationship of cerebellar purkinje cell simple spike discharge to movement kinematics in the monkey. *Journal of Neurophysiology*, 78, 1997.

[8] Donchin O, Gribova A, Steinberg O, Bergman H, and Vaadia E. Primary motor cortex is involved in bimanual coordination. *Nature*, 1998.

[9] Anthony G. Reina, Daniel W. Moran, and Andrew B. Schwartz. On the relationship between joint angular velocity and motor cortical discharge during reaching. *Journal of Neurophysiology*, 85:2576–2589, 2001.

[10] E. Vaadia, I. Haalman, M. Abeles, H. Bergman, Y. Prut, H. Slovin, and A. Aertsen. Dynamics of neuronal interactions in monkey cortex in relation to behavioral events. *Nature*, 373:515–518, Febuary 1995.

[11] Nicolelis MA Laubach M, Shuler M. Independent component analyses for quantifying neuronal ensemble interactions. *J Neurosci Methods*, 1999.

[12] A. Reihle, S. Grun, M. Diesmann, and A. M. H. J. Aersten. Spike synchronization and rate modulation differentially involved in motor cortical function. *Science*, 278:1950–1952, 1997.

[13] Chapin JK, Moxon KA, Markowitz RS, and Nicolelis MA. Real-time control of a robot arm using simultaneously recorded neurons in the motor cortex. *Nature Neuroscience*, 2:664–670, 1999.

[14] Miguel A. L. Nicolelis. Actions from thoughts. *Nature*, 409(18), January 2001.

[15] Johan Wessberg, Christopher R. Stambaugh, Jerald D. Kralik, Pamela D. Beck, Mark Laubach, John K. Chapin, Jung Kim, James Biggs, Mandayam A. Srinivasan, and Miguel A. L. Nicolelis. Real-time predictionof hand trajectory by ensembles of cortical neurons in primates. *Nature*, 408(16), November 2000.

[16] Nicolelis MA, Ghazanfar AA, Faggin BM, Votaw S, and Oliveira LM. Reconstructing the engram: simultaneous, multisite, many single neuron recordings. *Neuron*, 18:529–537, 1997.

[17] Isaacs RE, Weber DJ, and Schwartz A. Work toward real-time control of a cortical neural prothesis. *IEEE Trans Rehabil Eng*, 8(196–198), 2000.

[18] Dawn M. Taylor, Stephen I. Helms Tillery, and Andrew B. Schwartz. Direct cortical control of 3d neuroprosthetic devices. *Science*, 2002.

[19] Mijail D. Serruya, Nicholas G. Hatsopoulus, Liam Paninski, Matthew R. Fellows, and John P. Donoghue. Instant neural control of a movement signal. *Nature*, 416:141–142, March 2002.

[20] A. Smola and B. Sch. A tutorial on support vector regression, 1998.

[21] Vladimir Vapnik. *The Nature of Statistical Learning Theory*. Springer, N.Y., 1995.

[22] Tommi S. Jaakola and David Haussler. Exploiting generative models in discriminative calssifiers. In *NIPS*, 1998.

[23] Marc G. Genton. Classes of kernels for machine learning: A statistical perspective. *Journal of MAchine Learning Research*, 2:299–312, January 2001.

[24] Huma Lodhi, John Shawe-Taylor, Nello Cristianini, and Christopher J. C. H. Watkins. Text classification using string kernels. In *NIPS*, pages 563–569, 2000.
